# Automated Refinement of Bayes Networks' Parameters based on Test Ordering Constraints

**Omar Zia Khan & Pascal Poupart**
David R. Cheriton School of Computer Science
University of Waterloo
Waterloo, ON Canada
{ozkhan,ppoupart}@cs.uwaterloo.ca

**John Mark Agosta**[*]
Intel Labs
Santa Clara, CA, USA

johnmark.agosta@gmail.com

## Abstract

In this paper, we derive a method to refine a Bayes network diagnostic model by exploiting constraints implied by expert decisions on test ordering. At each step, the expert executes an evidence gathering test, which suggests the test's relative diagnostic value. We demonstrate that consistency with an expert's test selection leads to non-convex constraints on the model parameters. We incorporate these constraints by augmenting the network with nodes that represent the constraint likelihoods. Gibbs sampling, stochastic hill climbing and greedy search algorithms are proposed to find a MAP estimate that takes into account test ordering constraints and any data available. We demonstrate our approach on diagnostic sessions from a manufacturing scenario.

## 1 INTRODUCTION

The problem of learning-by-example has the promise to create strong models from a restricted number of cases; certainly humans show the ability to generalize from limited experience. Machine Learning has seen numerous approaches to learning task performance by imitation, going back to some of the approaches to inductive learning from examples [14]. Of particular interest are problem-solving tasks that use a model to infer the source, or cause of a problem from a sequence of investigatory steps or tests. The specific example we adopt is a diagnostic task such as appears in medicine, electro-mechanical fault isolation, customer support and network diagnostics, among others.

We define a *diagnostic sequence* as consisting of the assignment of values to a subset of tests. The *diagnostic process* embodies the choice of the best next test to execute at each step in the sequence, by measuring the *diagnostic value* among the set of available tests at each step, that is, the ability of a test to distinguish among the possible causes. One possible implementation with which to carry out this process, the one we apply, is a Bayes network [9]. As with all model-based approaches, provisioning an adequate model can be daunting, resulting in a "knowledge elicitation bottleneck."

A recent approach for easing the bottleneck grew out of the realization that the best time to gain an expert's insight into the model structure is during the diagnostic process. Recent work in "Query-Based Diagnostics" [1] demonstrated a way to improve model quality by merging model use and model building into a single process. More precisely the expert can take steps to modify the network structure to add or remove nodes or links, interspersed within the diagnostic sequence. In this paper we show how to extend this variety of learning-by-example to include also refinement of model parameters based on the expert's choice of test, from which we determine constraints. The nature of these constraints, as shown herein, is derived from the value of the tests to distinguish causes, a value referred to informally as *value of information* [10]. It is the effect of these novel constraints on network parameter learning that is elucidated in this paper.

---

[*]J. M. Agosta is no longer affiliated with Intel Corporation

Conventional statistical learning approaches are not suited to this problem, since the number of cases available from diagnostic sessions is small, and the data from any case is sparse. (Only a fraction of the tests are taken.) But more relevant is that one diagnostic sequence from an expert user represents the true behavior expected of the model, rather than a noisy realization of a case generated by the true model. We adopt a Bayesian approach, which offers a principled way to incorporate knowledge (constraints and data, when available) and also consider weakening the constraints, by applying a likelihood to them, so that possibly conflicting constraints can be incorporated consistently.

Sec. 2 reviews related work and Sec. 3 provides some background on diagnostic networks and model consistency. Then, Sec. 4 describes an augmented Bayesian network that incorporates constraints implied by an expert's choice of tests. Some sampling techniques are proposed to find the Maximum a posterior setting of the parameters given the constraints (and any data available). The approach is evaluated in Sec. 5 on synthetic data and a real world manufacturing diagnostic scenario. Finally, Sec. 6 discusses some future work.

## 2 RELATED WORK

Parameter learning for Bayesian networks can be viewed as searching in a high-dimensional space. Adopting constraints on the parameters based on some domain knowledge is a way of pruning this search space and learning the parameters more efficiently, both in terms of data needed and time required. Qualitative probabilistic networks [17] allow qualitative constraints on the parameter space to be specified by experts. For instance, the influence of one variable on another, or the combined influence of multiple variables on another variable [5] leads to linear inequalities on the parameters. Wittig and Jameson [18] explain how to transform the likelihood of violating qualitative constraints into a penalty term to adjust maximum likelihood, which allows gradient ascent and Expectation Maximization (EM) to take into account linear qualitative constraints.

Other examples of qualitative constraints include some parameters being larger than others, bounded in a range, within $\epsilon$ of each other, etc. Various proposals have been made that exploit such constraints. Altendorf et al. [2] provide an approximate technique based on constrained convex optimization for parameter learning. Niculescu et al. [15] also provide a technique based on constrained optimization with closed form solutions for different classes of constraints. Feelders [6] provides an alternate method based on isotonic regression while Liao and Ji [12] combine gradient descent with EM. de Campos and Ji [4] also use constrained convex optimization, however, they use Dirichlet priors on the parameters to incorporate any additional knowledge. Mao and Lebanon [13] also use Dirichlet priors, but they use probabilistic constraints to allow inaccuracies in the specification of the constraints.

A major difference between our technique and previous work is on the type of constraints. Our constraints do not need to be explicitly specified by an expert. Instead, we passively observe the expert and learn from what choices are made and not made [16]. Furthermore, as we shall show later, our constraints are non-convex, preventing the direct application of existing techniques that assume linear or convex functions. We use Beta priors on the parameters, which can easily be extended to Dirichlet priors like previous work. We incorporate constraints in an augmented Bayesian network, similar to Liang et al. [11], though their constraints are on model predictions as opposed to ours which are on the parameters of the network. Finally, we also use the notion of probabilistic constraints to handle potential mistakes made by experts.

## 3 BACKGROUND

### 3.1 DIAGNOSTIC BAYES NETWORKS

We consider the class of bipartite Bayes networks that are widely used as diagnostic models, though our approach can be used for networks with any structure. The network forms a sparse, directed, causal graph, where arcs go from causes to observable node variables. We use upper case to denote random variables; $C$ for causes, and $T$ for observables (tests). Lower case letters denote values in the domain of a variable, e.g. $c \in dom(C) = \{c, \bar{c}\}$, and bold letters denote sets of variables. A set of marginally independent binary-valued node variables $\mathbf{C}$ with distributions $\Pr(C)$ represent unobserved causes, and condition the remaining conditionally independent binary-valued test vari-

able nodes **T**. Each cause conditions one or more tests; likewise each test is conditioned by one or more causes, resulting in a graph with one or more possibly multiply-connected components. The test variable distributions $\Pr(T|C)$ incorporate the further modeling assumption of Independence of Causal Influence, the most familiar example being the Noisy-Or model [8]. To keep the exposition simple, we assume that all variables are binary and that conditional distributions are parametrized by the Noisy-Or; however, the algorithms described in the rest of the paper generalize to any discrete non-binary variable models.

Conventionally, unobserved tests are ranked in a diagnostic Bayes network by their Value Of Information (VOI) conditioned on tests already observed. To be precise, VOI is the expected gain in utility if the test were to be observed. The complete computation requires a model equivalent to a partially observable Markov decision process. Instead, VOI is commonly approximated by a greedy computation of the Mutual Information between a test and the set of causes [3]. In this case, it is easy to show that Mutual Information is in turn well approximated to second order by the *Gini impurity* [7] as shown in Equation 1.

$$\mathrm{GI}(\mathbf{C}|T) = \sum_t \Pr(T=t)\Big[\sum_{\mathbf{c}} \Pr(\mathbf{C}=\mathbf{c}|T=t)(1-\Pr(\mathbf{C}=\mathbf{c}|T=t))\Big] \tag{1}$$

We will use the Gini measure as a surrogate for VOI, as a way to rank the best next test in the diagnostic sequence.

## 3.2 MODEL CONSISTENCY

A model that is consistent with an expert would generate Gini impurity rankings consistent with the expert's diagnostic sequence. We interpret the expert's test choices as implying constraints on Gini impurity rankings between tests. To that effect, [1] defines the notion of *Cause Consistency* and *Test Consistency*, which indicate whether the cause and test orderings induced by the posterior distribution over causes and the VOI of each test agree with an expert's observed choice. Assuming that the expert greedily chooses the most informative test $T^*$ (i.e., test that yields the lowest Gini impurity) at each step, then the model is consistent with the expert's choices when the following constraints are satisfied:

$$\mathrm{GI}(\mathbf{C}|T^*) \leq \mathrm{GI}(\mathbf{C}|T_i) \qquad \forall i \tag{2}$$

We demonstrate next how to exploit these constraints to refine the Bayes network.

## 4 MODEL REFINEMENT

Consider a simple diagnosis example with two possible causes $C_1$ and $C_2$ and two tests $T_1$ and $T_2$ as shown in Figure 1. To keep the exposition simple, suppose that the priors for each cause are known (generally separate data is available to estimate these), but the conditional distribution of each test is unknown. Using the Noisy-OR parameterizations for the conditional distributions, the number of parameters are linear in the number of parents instead of exponential.

$$\Pr(T_i = true|\mathbf{C}) = 1 - (1-\theta_0^i)\prod_{j|C_j=true}(1-\theta_j^i) \tag{3}$$

Here, $\theta_0^i = \Pr(T_i = true|C_j = false\ \forall j)$ is the *leak probability* that $T_i$ will be true when none of the causes are true and $\theta_j^i = \Pr(T_i = true|C_j = true, C_k = false\ \forall k \neq j)$ is the *link reliability*, which indicates the independent contribution of cause $C_j$ to the probability that test $T_i$ will be true. In the rest of this section, we describe how to learn the $\theta$ parameters while respecting the constraints implied by test consistency.

### 4.1 TEST CONSISTENCY CONSTRAINTS

Suppose that an expert chooses test $T_1$ instead of test $T_2$ during the diagnostic process. This ordering by the expert implies that the current model (parametrized by the $\theta$'s) must be consistent with the constraint $\mathrm{GI}(\mathbf{C}|T_2) - \mathrm{GI}(\mathbf{C}|T_1) \geq 0$. Using the definition of Gini impurity in Eq. 1, we can rewrite

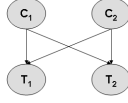

Figure 1: Network with 2 causes and 2 tests

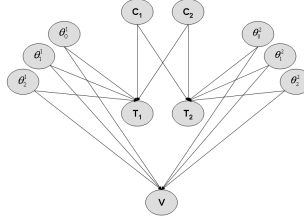

Figure 2: Augmented network with parameters and constraints

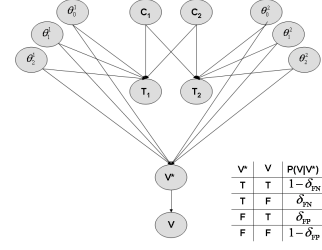

Figure 3: Augmented network extended to handle inaccurate feedback

the constraint for the network shown in Fig. 1 as follows:

$$\sum_{t_1}\left(\Pr(t_1)-\sum_{c_1,c_2}\frac{(\Pr(t_1|c_1,c_2)\Pr(c_1)\Pr(c_2))^2}{\Pr(t_1)}\right)-\sum_{t_2}\left(\Pr(t_2)-\sum_{c_1,c_2}\frac{(\Pr(t_2|c_1,c_2)\Pr(c_1)\Pr(c_2))^2}{\Pr(t_2)}\right)\geq 0 \tag{4}$$

Furthermore, using the Noisy-Or encoding from Eq. 3, we can rewrite the constraint as a polynomial in the $\theta$'s. This polynomial is non-linear, and in general, not concave. The feasible space may consist of disconnected regions. Fig. 4 shows the surface corresponding to the polynomial for the case where $\theta_0^i = 0$ and $\theta_1^i = 0.5$ for each test $i$, which leaves $\theta_2^1$ and $\theta_2^2$ as the only free variables. The parameters' feasible space, satisfying the constraint consists of the two disconnected regions where the surface is positive.

## 4.2 AUGMENTED BAYES NETWORK

Our objective is to learn the $\theta$ parameters of diagnostic Bayes networks given test constraints of the form described in Eq. 4. To deal with non-convex constraints and disconnected feasible regions, we pursue a Bayesian approach whereby we explicitly model the parameters and constraints as random variables in an augmented Bayes network (see Fig. 2). This allows us to frame the problem of learning the parameters as an inference problem in a hybrid Bayes network of discrete ($\mathbf{T}, \mathbf{C}, V$) and continuous ($\boldsymbol{\Theta}$) variables. As we will see shortly, this augmented Bayes network provides a unifying framework to simultaneously learn from constraints and data, to deal with possibly inconsistent constraints, and to express preferences over the degree of satisfaction of the constraints.

We encode the constraint derived from the expert feedback as a binary random variable $V$ in the Bayes network. If $V$ is $true$ the constraint is satisfied, otherwise it is violated. Thus, if $V$ is $true$ then $\boldsymbol{\Theta}$ lies in the positive region of Fig. 4, and if $V$ is $false$ then $\boldsymbol{\Theta}$ lies in the negative region. We model the CPT for $V$ as $\Pr(V|\boldsymbol{\Theta}) = \max(0,\pi)$, where $\pi = \mathrm{GI}(\mathbf{C}|T_1) - \mathrm{GI}(\mathbf{C}|T_2)$. Note that the value of $\mathrm{GI}(\mathbf{C}|T)$ lies in the interval [0,1], so the probability $\pi$ will always be normalized. The intuition behind this definition of the CPT for $V$ is that a constraint is more likely to be satisfied if the parameters lie in the interior of the constraint region.

We place a Beta prior over each $\Theta$ parameter. Since the test variables are conditioned on the $\Theta$ parameters that are now part of the network, their conditional distributions become known. For instance, the conditional distribution for $T_i$ (given in Eq. 3) is fully defined given the noisy-or parameters $\theta_j^i$. Hence the problem of learning the parameters becomes an inference problem to compute posteriors over the parameters given that the constraint is satisfied (and any data). In practice, it is more convenient to obtain a single value for the parameters instead of a posterior distribution since it is easier to make diagnostic predictions based on one Bayes network. We estimate the parameters by computing a maximum a posteriori (MAP) hypothesis given that the constraint is satisfied (and any data): $\boldsymbol{\Theta}^* = \arg\max_{\boldsymbol{\Theta}} \Pr(\boldsymbol{\Theta}|V = true)$.

---

**Algorithm 1** Pseudo Code for Gibbs Sampling, Stochastic Hill Climbing and Greedy Search

---

1    Fix observed variables, let $V = true$ and randomly sample feasible starting state $\mathbf{S}$
2    **for** $i = 1$ **to** $\#samples$
3        **for** $j = 1$ **to** $\#hiddenVariables$
4            $acceptSample = false; k = 0$
5            **repeat**
6                Sample $s'$ from conditional of $j^{th}$ hidden variable $S_j$
7                $\mathbf{S}' = \mathbf{S}; S_j = s'$
8                **if** $S_j$ is cause or test, then $acceptSample = true$
9                **elseif** $\mathbf{S}'$ obeys constraints $\mathbf{V}^*$
10                   **if** algo == Gibbs
11                       Sample $u$ from uniform distribution, U(0,1)
12                       **if** $u < \frac{p(\mathbf{S}')}{Mq(\mathbf{S}')}$ where $p$ and $q$ are the true and proposal distributions and $M > 1$
13                           $acceptSample = true$
14                   **elseif** algo == StochasticHillClimbing
15                       **if** $likelihood(\mathbf{S}') > likelihood(\mathbf{S})$, then $acceptSample = true$
16                   **elseif** algo == Greedy, then $acceptSample = true$
17                **elseif** algo == Greedy
18                    $k = k + 1$
19                    **if** $k$ == $maxIterations$, then $s' = S_j; acceptSample = true$
20            **until** $acceptSample$ == $true$
21            $S_j = s'$

---

## 4.3   MAP ESTIMATION

Previous approaches for parameter learning with domain knowledge include modified versions of EM or some other optimization techniques that account for linear/convex constraints on the parameters. Since our constraints are non-convex, we propose a new approach based on Gibbs sampling to approximate the posterior distribution, from which we compute the MAP estimate. Although the technique converges to the MAP in the limit, it may require excessive time. Hence, we modify Gibbs sampling to obtain more efficient stochastic hill climbing and greedy search algorithms with anytime properties.

The pseudo code for our Gibbs sampler is provided in Algorithm 1. The two key steps are sampling the conditional distributions of each variable (line 6) and rejection sampling to ensure that the constraints are satisfied (lines 9 and 12). We sample each variable given the rest according to the following distributions:

$$t_i \sim \Pr(T_i|\mathbf{c}, \theta_\mathbf{i}) \;\; \forall i \tag{5}$$

$$c_j \sim \Pr(C_j|\mathbf{c} - c_j, \mathbf{t}, \theta) \propto \prod_j \Pr(C_j) \prod_i \Pr(t_i|\mathbf{c}, \theta_\mathbf{i}) \;\; \forall j \tag{6}$$

$$\theta_j^i \sim \Pr(\Theta_j^i|\mathbf{\Theta} - \Theta_j^i, \mathbf{t}, \mathbf{c}, v) \propto \Pr(v|\mathbf{t}, \mathbf{\Theta}) \prod_i \Pr(t_i|\mathbf{c}_j, \theta_i) \;\; \forall i, j \tag{7}$$

The tests and causes are easily sampled from the multinomials as described in the equations above. However, sampling the $\theta$'s is more difficult due to the factor $\Pr(v|\mathbf{\Theta}, \mathbf{t}) = \max(0, \pi)$, which is a truncated mixture of Betas. So, instead of sampling $\theta$ from its true conditional, we sample it from a proposal distribution that replaces $\max(0, \pi)$ by an un-truncated mixture of Betas equal to $\pi + a$ where $a$ is a constant that ensures that $\pi + a$ is always positive. This is equivalent to ignoring the constraints. Then we ensure that the constraints are satisfied by rejecting the samples that violate the constraints. Once Gibbs sampling has been performed, we obtain a sample that approximates the posterior distribution over the parameters given the constraints (and any data). We return a single setting of the parameters by selecting the sampled instance with the highest posterior probability (i.e., MAP estimate). Since we will only return the MAP estimate, it is possible to speed up the search by modifying Gibbs sampling. In particular, we obtain a stochastic hill climbing algorithm by accepting a new sample only if its posterior probability improves upon that of the previous sample

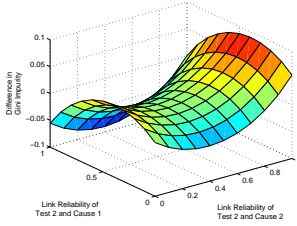
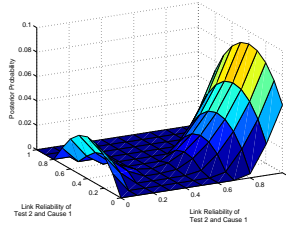
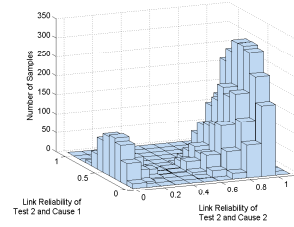

Figure 4: Difference in Gini impurity for the network in Fig. 1 when $\theta_2^1$ and $\theta_2^2$ are the only parameters allowed to vary.

Figure 5: Posterior over parameters computed through calculation after discretization.

Figure 6: Posterior over parameters calculated through Sampling.

(line 15). Thus, each iteration of the stochastic hill climber requires more time, but always improves the solution.

As the number of constraints grows and the feasibility region shrinks, the Gibbs sampler and stochastic hill climber will reject most samples. We can mitigate this by using a Greedy sampler that caps the number of rejected samples, after which it abandons the sampling for the current variable to move on to the next variable (line 19). Even though the feasibility region is small overall, it may still be large in some dimensions, so it makes sense to try sampling another variable (that may have a larger range of feasible values) when it is taking too long to find a new feasible value for the current variable.

### 4.4  MODEL REFINEMENT WITH INCONSISTENT CONSTRAINTS

So far, we have assumed that the expert's actions generate a feasible region as a consequence of consistent constraints. We handle inconsistencies by further extending our augmented diagnostic Bayes network. We treat the observed constraint variable, $V$, as a probabilistic indicator of the true constraint $V^*$ as shown in Figure 3. We can easily extend our techniques for computing the MAP to cater for this new constraint node by sampling an extra variable.

## 5  EVALUATION AND EXPERIMENTS

### 5.1  EVALUATION CRITERIA

Formally, for $M^*$, the true model that we aim to learn, the diagnostic process determines the choice of best next test as the one with the smallest Gini impurity. If the correct choice for the next test is known (such as demonstrated by an expert), we can use this information to include a constraint on the model. We denote by $\mathbf{V}^+$ the set of observed constraints and by $\mathbf{V}^*$ the set of all possible constraints that hold for $M^*$. Having only observed $\mathbf{V}^+$, our technique will consider any $M^+ \in \mathbf{M}^+$ as a possible true model, where $\mathbf{M}^+$ is the set of all models that obey $V^+$. We denote by $\mathbf{M}^*$ the set of all models that are *diagnostically equivalent* to $M^*$ (i.e., obey $V^*$ and would recommend the same steps as $M^*$) and by $M_{\mathbf{V}^+}^{\mathrm{MAP}}$ the particular model obtained by MAP estimation based on the constraints $\mathbf{V}^+$. Similarly, when a dataset $\mathcal{D}$ is available, we denote by $M_{\mathbf{D}}^{\mathrm{MAP}}$ the model obtained by MAP estimation based on $\mathbf{D}$ and by $M_{\mathbf{DV}^+}^{\mathrm{MAP}}$, the model based on $\mathbf{D}$ and $\mathbf{V}^+$.

Ideally we would like to find the true underlying model $M^*$, hence we will report the KL divergence between the models found and $M^*$. However, other *diagnostically equivalent* $M^*$ may recommend the same tests as $M^*$ and thus have similar constraints, so we also report *test consistency* with $M^*$ (i.e., # of recommended tests that are the same).

### 5.2  CORRECTNESS OF MODEL REFINEMENT

Given $\mathbf{V}^*$, our technique for model adjustment is guaranteed to choose a model $M^{\mathrm{MAP}} \in \mathbf{M}^*$ by construction. If any constraint $V^* \in \mathbf{V}^*$ is violated, the rejection sampling step of our technique

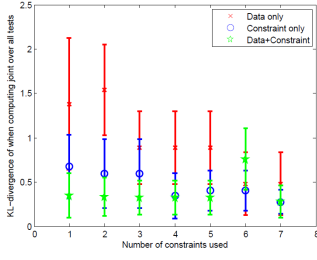

Figure 7: Mean KL-divergence and one standard deviation for a 3 cause 3 test network on learning with data, constraints and data+constraints.

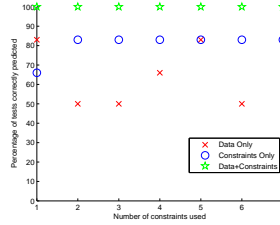

Figure 8: Test Consistency for a 3 cause 3 test network on learning with data, constraints and data+constraints.

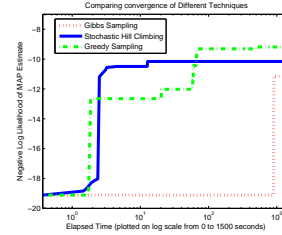

Figure 9: Convergence rate comparison.

would reject that set of parameters. To illustrate this, consider the network in Fig. 2. There are six parameters (four link reliabilities and two leak parameters). Let us fix the leak parameters and the link reliability from the first cause to each test. Now we can compute the posterior surface over the two variable parameters after discretizing each parameter in small steps and then calculating the posterior probability at each step as shown in Fig. 5. We can compare this surface with that obtained after Gibbs sampling using our technique as shown in Fig. 6. We can see that our technique recovers the posterior surface from which we can compute the MAP. We obtain the same MAP estimate with the stochastic hill climbing and greedy search algorithms.

## 5.3 EXPERIMENTAL RESULTS ON SYNTHETIC PROBLEMS

We start by presenting our results on a 3-cause by 3-test fully-connected bipartite Bayes network. We assume that there exists some $M^* \in \mathbf{M}^*$ that we want to learn given $\mathbf{V}^+$. We use our technique to find $M^{\text{MAP}}$. To evaluate $M^{\text{MAP}}$, we first compute the constraints, $\mathbf{V}^*$ for $M^*$ to get the feasible region associated with the true model. Next, we sample 100 other models from this feasible region that are diagnostically equivalent. We compare these models with $M^{\text{MAP}}$ (after collecting 200 samples with non-informative priors for the parameters).

We compute the KL-divergence of $M^{\text{MAP}}$ with respect to each sampled model. We expect KL-divergence to decrease as the number of constraints in $\mathbf{V}^+$ increases since the feasible region becomes smaller. Figure 7 confirms this trend and shows that $M^{\text{MAP}}_{\text{DV}+}$ has lower mean KL-divergence than $M^{\text{MAP}}_{\mathbf{V}+}$, which has lower mean KL-divergence than $M^{\text{MAP}}_{\mathbf{D}}$. The data points in $\mathbf{D}$ are limited to the results of the diagnostic sessions needed to obtain $\mathbf{V}^+$. As constraints increase, more data is available and so the results for the data-only approach also improve with increasing constraints.

We also compare the test consistency when learning from data only, constraints only or both. Given a fixed number of constraints, we enumerate the unobserved trajectories, and then compute the highest ranked test using the learnt model and the sampled true models, for each trajectory. The test consistency is reported as a percentage, with 100% consistency indicating that the learned and true models had the same highest ranked tests on every trajectory. Figure 8 presents these percentatges for the greedy sampling technique (the results are similar for the other techniques). It again appears that learning parameters with both constraints and data is better than learning with only constraints, which is most of the times better than learning with only data.

Figure 9 compares the convergence rate of each technique to find the MAP estimate. As expected, Stochastic Hill Climbing and Greedy Sampling take less time than Gibbs sampling to find parameter settings with high posterior probability.

## 5.4 EXPERIMENTAL RESULTS ON REAL-WORLD PROBLEMS

We evaluate our technique on a real-world diagnostic network collected and reported by Agosta et al. [1], where the authors collected detailed session logs over a period of seven weeks in which the

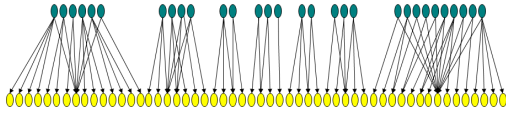

Figure 10: Diagnostic Bayesian network collected from user trials and pruned to retain sub-networks with at least one constraint

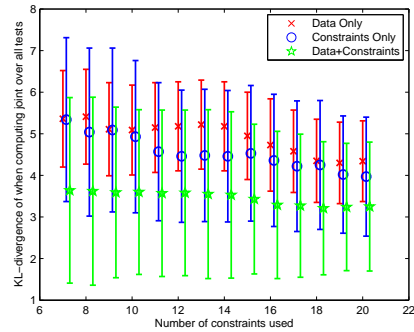

Figure 11: KL divergence comparison as the number of constraints increases for the real world problem.

entire diagnostic sequence was recorded. The sequences intermingle model building and querying phases. The model network structure was inferred from an expert's sequence of positing causes and tests. Test-ranking constraints were deduced from the expert's test query sequences once the network structure is established.

The 157 sessions captured over the seven weeks resulted in a Bayes network with 115 tests, 82 root causes and 188 arcs. The network consists of several disconnected sub-networks, each identified with a symptom represented by the first test in the sequence, and all subsequent tests applied within the same subnet. There were 20 sessions from which we were able to observe trajectories with at least two tests, resulting in a total of 32 test constraints. We pruned our diagnostic network to remove the sub-networks with no constraints to get a Bayes network with 54 tests, 30 root causes, and 67 parameters divided in 7 sub-networks, as shown in Figure 10, on which we apply our model refinement technique to learn the parameters for each sub-network separately.

Since we don't have the true underlying network and the full set of constraints (more constraints could be observed in future diagnostic sessions), we treated the 32 constraints as if they were $\mathbf{V}^*$ and the corresponding feasible region $\mathbf{M}^*$ as if it contained models diagnostically equivalent to the unknown true model. Figure 11 reports the KL divergence between the models found by our algorithms and sampled models from $\mathbf{M}^*$ as we increase the number of constraints. With such limited constraints and consequently large feasible regions, it is not surprising that the variation in KL divergence is large. Again, the MAP estimate based on both the constraints and the data has lower KL divergence than constraints only and data only.

## 6   CONCLUSION AND FUTURE WORK

In summary, we presented an approach that can learn the parameters of a Bayes network based on constraints implied by test consistency and any data available. While several approaches exist to incorporate qualitative constraints in learning procedures, our work makes two important contributions: First, this is the first approach that exploits implicit constraints based on value of information assessments. Secondly it is the first approach that can handle non-convex constraints. We demonstrated the approach on synthetic data and on a real-world manufacturing diagnostic problem. Since data is generally sparse in diagnostics, this work makes an important advance to mitigate the model acquisition bottleneck, which has prevented the widespread application of diagnostic networks so far. In the future, it would be interesting to generalize this work to reinforcement learning in applications where data is sparse, but constraints may be inferred from expert interactions.

### Acknowledgments

This work was supported by a grant from Intel Corporation.

# References

[1] John Mark Agosta, Omar Zia Khan, and Pascal Poupart. Evaluation results for a query-based diagnostics application. In *The Fifth European Workshop on Probabilistic Graphical Models (PGM 10)*, Helsinki, Finland, September 13–15 2010.

[2] Eric E. Altendorf, Angelo C. Restificar, and Thomas G. Dietterich. Learning from sparse data by exploiting monotonicity constraints. In *Proceedings of Twenty First Conference on Uncertainty in Artificial Intelligence (UAI)*, Edinburgh, Scotland, July 2005.

[3] Brigham S. Anderson and Andrew W. Moore. Fast information value for graphical models. In *Proceedings of Nineteenth Annual Conference on Neural Information Processing Systems (NIPS)*, pages 51–58, Vancouver, BC, Canada, December 2005.

[4] Cassio P. de Campos and Qiang Ji. Improving Bayesian network parameter learning using constraints. In *International Conference in Pattern Recognition (ICPR)*, Tampa, FL, USA, 2008.

[5] Marek J. Druzdzel and Linda C. van der Gaag. Elicitation of probabilities for belief networks: combining qualitative and quantitative information. In *Proceedings of the Eleventh Annual Conference on Uncertainty in Artificial Intelligence (UAI)*, pages 141–148, Montreal, QC, Canada, 1995.

[6] Ad J. Feelders. A new parameter learning method for Bayesian networks with qualitative influences. In *Proceedings of Twenty Third International Conference on Uncertainty in Artificial Intelligence (UAI)*, Vancouver, BC, July 2007.

[7] Mara Angeles Gil and Pedro Gil. A procedure to test the suitability of a factor for stratification in estimating diversity. *Applied Mathematics and Computation*, 43(3):221 – 229, 1991.

[8] David Heckerman and John S. Breese. Causal independence for probability assessment and inference using bayesian networks. *IEEE Systems, Man, and Cybernetics*, 26(6):826–831, November 1996.

[9] David Heckerman, John S. Breese, and Koos Rommelse. Decision-theoretic troubleshooting. *Communications of the ACM*, 38(3):49–56, 1995.

[10] Ronald A. Howard. Information value theory. *IEEE Transactions on Systems Science and Cybernetics*, 2(1):22–26, August 1966.

[11] Percy Liang, Michael I. Jordan, and Dan Klein. Learning from measurements in exponential families. In *Proceedings of Twenty Sixth Annual International Conference on Machine Learning (ICML)*, Montreal, QC, Canada, June 2009.

[12] Wenhui Liao and Qiang Ji. Learning Bayesian network parameters under incomplete data with domain knowledge. *Pattern Recognition*, 42:3046–3056, 2009.

[13] Yi Mao and Guy Lebanon. Domain knowledge uncertainty and probabilistic parameter constraints. In *Proceedings of Twenty Fifth Conference on Uncertainty in Artificial Intelligence (UAI)*, Montreal, QC, Canada, 2009.

[14] Ryszard S. Michalski. A theory and methodology of inductive learning. *Artificial Intelligence*, 20:111–116, 1984.

[15] Radu Stefan Niculescu, Tom M. Mitchell, and R. Bharat Rao. Bayesian network learning with parameter constraints. *Journal of Machine Learning Research*, 7:1357–1383, 2006.

[16] Mark A. Peot and Ross D. Shachter. Learning from what you dont observe. In *Proceedings of the Fourteenth Conference on Uncertainty in Artificial Intelligence (UAI)*, pages 439–446, Madison, WI, July 1998.

[17] Michael P. Wellman. Fundamental concepts of qualitative probabilistic networks. *Artificial Intelligence*, 44(3):257–303, August 1990.

[18] Frank Wittig and Anthony Jameson. Exploiting qualitative knowledge in the learning of conditional probabilities of Bayesian networks. In *Proceedings of the Sixteenth Conference on Uncertainty in Artificial Intelligence (UAI)*, San Francisco, CA, July 2000.

